# Interactive Deep Clustering via Value Mining

**Honglin Liu[1], Peng Hu[1], Changqing Zhang[2,3], Yunfan Li[1]\*, Xi Peng[1,4]\***

[1]College of Computer Science, Sichuan University, Chengdu, China
[2]College of Intelligence and Computing, Tianjin University, Tianjin, China
[3]Tianjin Key Lab of Machine Learning, Tianjin, China
[4]State Key Laboratory of Hydraulics and Mountain River Engineering,
Sichuan University, Chengdu, China
{tristanliuhl, penghu.ml, yunfanli.gm, pengx.gm}@gmail.com,
zhangchangqing@tju.edu.cn

## Abstract

In the absence of class priors, recent deep clustering methods resort to data augmentation and pseudo-labeling strategies to generate supervision signals. Though achieved remarkable success, existing works struggle to discriminate hard samples at cluster boundaries, mining which is particularly challenging due to their unreliable cluster assignments. To break such a performance bottleneck, we propose incorporating user interaction to facilitate clustering instead of exhaustively mining semantics from the data itself. To be exact, we present Interactive Deep Clustering (IDC), a plug-and-play method designed to boost the performance of pre-trained clustering models with minimal interaction overhead. More specifically, IDC first quantitatively evaluates sample values based on hardness, representativeness, and diversity, where the representativeness avoids selecting outliers and the diversity prevents the selected samples from collapsing into a small number of clusters. IDC then queries the cluster affiliations of high-value samples in a user-friendly manner. Finally, it utilizes the user feedback to finetune the pre-trained clustering model. Extensive experiments demonstrate that IDC could remarkably improve the performance of various pre-trained clustering models, at the expense of low user interaction costs. The code could be accessed at pengxi.me.

## 1 Introduction

Clustering aims at partitioning samples into semantically distinct groups. In recent years, deep clustering methods [5, 30, 15, 41, 13], powered by the feature extraction ability of neural networks, have excelled in handling large-scale and high-dimensional data across various domains, including image segmentation [7], anomaly detection [25], medical analysis [1], bioinformatics [20], and so on.

To discover the semantical data partitions, the core of deep clustering lies in designing supervision signals to extract discriminative information from data. To this end, early efforts reformulate the self-representation property [31], hierarchical structure [43], or assignment distribution prior [42] into differentiable objectives for model optimization. Recently, inspired by the success of contrastive learning [6, 11], the community has shifted towards constructing self-supervision signals via data augmentations, thus promoting the contrastive clustering paradigm [18, 46, 14]. The latest research indicates that pseudo-labels could further enhance the clustering performance [38, 21, 29, 22].

Despite these merits, almost all deep clustering methods suffer from the performance ceiling due to the limited information inherent in the data [19]. Particularly, this limitation is reflected in the poor discrimination of hard boundary samples as shown in Fig. 1a. Consequently, it has a great chance to

---

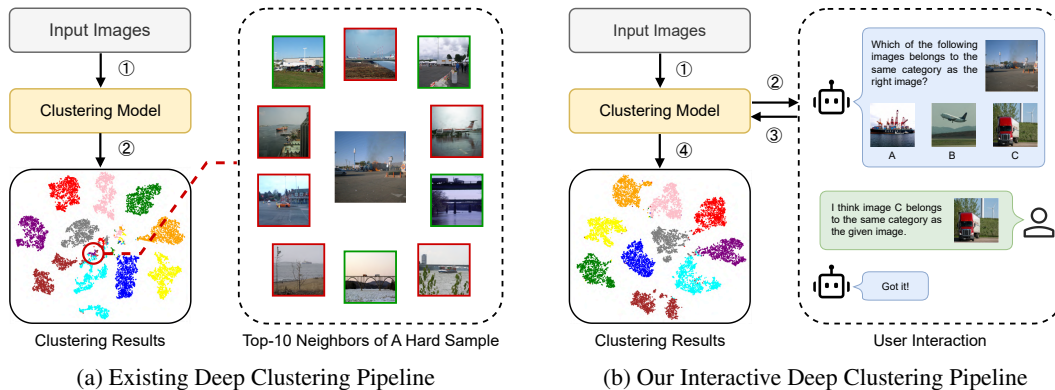

(a) Existing Deep Clustering Pipeline      (b) Our Interactive Deep Clustering Pipeline

Figure 1: Our key idea. (a) Existing deep clustering methods suffer from poor discrimination of hard boundary samples. As a showcase, we highlight one hard sample (red circle) whose neighborhood includes visually similar but semantically different neighbors (red boxes), leading to a performance bottleneck. (b) Instead of exhaustively mining internal semantics from data, we propose incorporating external user interaction to address the hard sample problem. In brief, we select high-value samples and query their cluster affiliations, which improves the clustering performance remarkably as visualized in the T-SNE plots.

improve the overall performance remarkably through mining hard samples. Current pseudo-labeling strategies, however, focus on easy samples with high-confident cluster predictions, while failing to handle hard boundary samples with unreliable predictions. To tackle hard samples, a recent effort attempts to mitigate their impact by neglecting them when constructing neighborhoods [45]. Nevertheless, this approach, akin to an ostrich avoidance policy, essentially sidesteps the core problem rather than solving it, ultimately leaving hard samples inseparable.

Acknowledging limitations in tackling hard samples internally, we present a straightforward approach by incorporating external user interaction, as illustrated in Fig. 1b. In brief, given an arbitrary pre-trained clustering model, we aim to correct its cluster assignments of hard samples with minimal user interaction overhead. To achieve this, we confront two challenges: *i*) constructing an efficient and user-friendly interaction interface, and *ii*) effectively utilizing user feedback. To tackle the first challenge, we present a novel strategy to mine valuable samples based on hardness, representativeness, and diversity for user inquiries with mathematical formulations. Here, the representativeness is designed to avoid selecting outliers and the diversity is used to prevent the selected samples from collapsing into a small number of clusters. For user convenience, instead of directly requesting class labels, we inquire about the affiliation of each selected sample with its nearest cluster centers. For the second challenge, we design two new losses to finetune the pre-trained model using both positive and negative user feedback. Specifically, positive feedback indicates the semantic alignment w.r.t. the selected cluster, while negative feedback denies all candidate clusters as semantically inconsistent. Additionally, we propose a regularization loss to preserve the overall cluster boundary of the original model, preventing it from overfitting the inquired samples. Notably, our method is a model-irrelevant plug-in that can be effortlessly integrated into existing clustering methods, thereby enhancing their performance.

The major contributions of this work could be summarized as follows:

- We propose incorporating external user interaction to break through the performance ceiling of existing deep clustering methods, specifically correcting the hard samples that are indistinguishable internally.

- To reduce interaction costs, we present a value-mining strategy to select hard, representative, and diverse samples for user inquiry. To simplify the interaction, we design a user-friendly interface to ask for cluster affiliations of these selected valuable samples.

- The proposed IDC could be easily integrated into any pre-trained deep clustering model. Extensive experiments demonstrate that our IDC could significantly boost the performance of various state-of-the-art deep clustering methods with negligible user interaction costs.

## 2 Related Work

In this section, we briefly review two fields related to this work, namely, deep clustering and hard sample mining.

### 2.1 Deep Clustering

Thanks to the powerful feature extraction ability, deep clustering methods have shown promising results on complex real-world data and advanced rapidly in past years [42, 43, 5, 15]. Recently, the success of self-supervised learning [6, 11, 10] gives rise to a series of contrastive clustering methods [18, 46, 21, 14]. However, even enhanced by data augmentation and pseudo-labeling strategies [38, 29, 21, 4], the performance of existing deep clustering methods is inherently upper-bounded by the limited internal supervision signals. Instead of exhaustively mining semantics from the data, a recent work attempts to leverage external data and models to facilitate clustering [19]. Another branch of study focuses on integrating prior class labels [3, 16] or pairwise constraints [39, 40, 24, 27] into the clustering process to boost the performance.

Different from existing studies that pursue overall performance improvements, this work aims to address the specific hard sample problem. Notably, the performance bottleneck of existing methods lies in the poor discrimination for hard samples at the cluster boundaries. Given the difficulty of internally correcting cluster assignments for hard boundary samples, we propose incorporating external user interaction as a straightforward solution. By inquiring about the cluster affiliations of representative and diverse hard samples, our method could significantly boost the performance of pre-trained clustering models with low interaction overhead.

### 2.2 Hard Sample Mining

Hard samples refer to data points that are difficult to recognize and understand due to their ambiguous or weak semantics, which widely exist in various tasks such as face recognition [33], person re-identification [44], image segmentation [28], object detection [2], and cross-modal retrieval [23]. On the one hand, the model is likely to make wrong predictions for these samples. On the other hand, mining these samples could significantly improve the model performance. Notably, hard sample mining is usually conducted in a supervised manner. For clustering, it is daunting to correct the assignments of hard samples at cluster boundaries by the model itself due to the absence of class label priors. As an attempt, SeCu [32] recently proposes assigning larger weights to hard samples when computing cluster centers for better cluster discriminability. However, the improvement is limited due to the unreliable cluster assignments of hard samples.

The differences between this work and previous hard sample mining methods are twofold. On the one hand, most existing works focus on enclosed supervised learning, while we explore hard sample mining for unsupervised clustering by incorporating user interaction. On the other hand, unlike previous works that solely pursue sample hardness, we further consider the representativeness and diversity of hard samples, resulting in a more comprehensive evaluation of data value. Such a value mining strategy helps to improve the cluster model with interaction costs as low as possible.

## 3 Method

In this section, we introduce our novel Interactive Deep Clustering method (IDC). As illustrated in Fig. 2, IDC consists of two primary stages: user interaction and model optimization. Initially, IDC solicits the user to determine the cluster affiliation of highly valuable samples, which are strategically selected based on their **hardness**, **representativeness**, and **diversity**. Subsequently, during the model optimization stage, IDC refines the cluster assignments of these samples according to the user feedback, while preserving the overall decision boundary of the pre-trained model. The two stages are further detailed in Sections 3.1 and 3.2.

### 3.1 User Interaction

To boost the pre-trained clustering model with minimal user interaction cost, we select the most valuable samples for user inquiries. The value of each sample is appraised based on three proposed

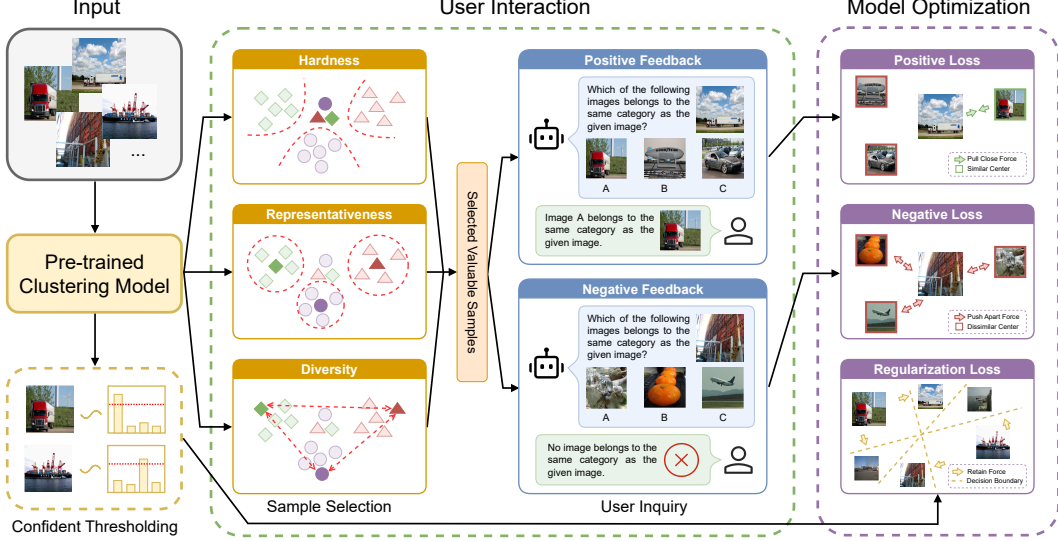

Figure 2: The overall framework of IDC consists of two stages: user interaction and model optimization. In the user interaction stage, given a pre-trained clustering model, IDC first selects high-value samples based on hardness, representativeness, and diversity. Then it inquires the user about the affiliations of the selected samples relative to their nearest cluster centers. In the model optimization stage, IDC utilizes both positive and negative user feedback to finetune the pre-trained model with positive and negative losses for cluster performance improvement. Meanwhile, IDC adopts a regularization loss on high-confident predictions to prevent overfitting inquired samples.

criteria: hardness $h$, representativeness $r$, and diversity $d$, encapsulated by the equation:

$$v_i = h_i + r_i + d_i, \tag{1}$$

where $v_i$ denotes the importance of the $i$-th sample. We elaborate on the three metrics as follows:

**Hardness.** Typically, a pre-trained clustering model could accurately assign clusters to easy samples near cluster centers. However, it may fail on hard samples situated at cluster peripheries. In other words, identifying these boundary samples is pivotal for boosting the clustering performance. Therefore, we quantify the hardness of the $i$-th sample by its proximity to cluster centers:

$$h_i = \log(1 - z_i \cdot c_{g_1} + z_i \cdot c_{g_2}), \tag{2}$$

where $z_i$ is the L2-normalized feature of the $i$-th sample, and $c_{g_1}, c_{g_2}$ denote the closest and second-closet cluster centers to $z_i$, respectively. A higher $h_i$ score indicates greater uncertainty in cluster assignment for the $i$-th sample.

**Representativeness.** While correcting hard samples is beneficial, focusing solely on hardness may lead to suboptimal results, as the most challenging samples could be outliers that negatively impact the model's generalization ability. To tackle this problem, we prefer samples reside in dense regions, where inquiring about a single sample could correct the cluster assignments of numerous adjacent ones. Formally, we define the representativeness of the $i$-th sample by the density of its $K$ nearest neighbors as follows:

$$r_i = -\log \sum_{j=1}^{K} \|z_i - z_{i(j)}\|_2^2, \tag{3}$$

where $z_{i(j)}$ refers to the $j$-th nearest neighbor of $z_i$, and $K$ is the number of nearest neighbors empirically set to 20. A higher $r_i$ score suggests a more compact local structure, indicating that the $i$-th sample is more representative.

**Diversity.** In practice, we discover that pursuing hardness and representativeness may result in an unbalanced sample distribution, heavily collapsing into a small number of clusters as shown in Fig. 3. To avoid this, we present the "diversity" metric to ensure sufficient dispersion of the selected samples. Different from hardness and representativeness which are independent of the selection,

---

**Algorithm 1** Valuable Sample Selection

---

**Input:** Sample features $\mathbf{Z} = \{\mathbf{z}_1, \ldots, \mathbf{z}_N\}$, number of samples to be selected $M$
**Output:** Selected sample indices $\mathbf{S} = \{s_1, \ldots, s_M\}$
 1: Initialize the selected indices $\mathbf{S} = \{\}$ and the remaining indices $\mathbf{R} = \{1, \ldots, N\}$
 2: Compute cluster centers of $\mathbf{Z}$ by $k$-means
 3: **for** $i \in [1, N]$ **do**
 4:     Compute the hardness score $h_i$ by Eq. (2)
 5:     Compute the representativeness score $r_i$ by Eq. (3)
 6:     Initialize the diversity score $d_i = 0$, since no sample has been selected
 7:     Compute the value score $v_i$ by Eq. (1)
 8: **end for**
 9: **for** $j \in [1, M]$ **do**
10:     Select the $s_j$-th sample with the highest value from $\mathbf{Z_R}$
11:     $\mathbf{S} = \mathbf{S} \cup \{s_j\}, \mathbf{R} = \mathbf{R} \setminus \{s_j\}$
12:     Update the diversity score $d$ for $\mathbf{Z_R}$ by Eq. (4)
13:     Update the value score $v$ for $\mathbf{Z_R}$ by Eq. (1)
14: **end for**

---

the diversity of a given sample is measured by its deviation from previously selected samples. In our implementation, the sample with the highest $v_i$ score is selected iteratively until $M$ samples are selected. In each iteration, the diversity of the $i$-th sample is computed according to the already selected samples:

$$d_i = \min_{j \in \mathbf{S}} \ \log(1 - z_i \cdot z_j), \tag{4}$$

where $\mathbf{S}$ represents the indices of the selected samples.

For user interaction, we select the top $M = 500$ valuable samples with the highest $v_i$ scores in our experiments. The selection process is outlined in Algorithm 1. According to Theorem 1 proved below, IDC could select the most valuable samples to minimize the user interaction cost.

**Theorem 1.** *The value of the selected sample decreases as the selection progresses, i.e.,*

$$v_{s_j}^j \geq v_{s_{j+1}}^{j+1}, \forall j \in [1, M-1]. \tag{5}$$

*where $s_j$ denotes the index of the $j$-th selected sample, and $v_i^j$ denotes the $i$-th sample's value in the $j$-th selection.*

***Proof.*** We denote $\mathbf{S}^j$ as the set of selected sample indices and $\mathbf{R}^j$ as the set of remaining sample indices after the $j$-th selection. Further, the $i$-th sample's hardness, representativeness, and diversity in the $j$-th selection (*i.e.*, $i \in \mathbf{R}^{j-1}$) are denoted as $h_i^j$, $r_i^j$, and $d_i^j$, respectively. By the definition of sample value in Eq. (1), we have

$$v_i^j = h_i^j + r_i^j + d_i^j, \tag{6}$$

Notably, since we choose $s_j$ instead of $s_{j+1}$ in the $j$-th selection, there must be

$$v_{s_j}^j \geq v_{s_{j+1}}^j. \tag{7}$$

By the definition of diversity in Eq. (4), we have

$$d_{s_{j+1}}^j = \min_{i \in \mathbf{S}^{j-1}} \log(1 - z_{s_{j+1}} \cdot z_i) \geq \min_{i \in \mathbf{S}^j} \log(1 - z_{s_{j+1}} \cdot z_i) = d_{s_{j+1}}^{j+1}, \tag{8}$$

where the inequality holds since $\mathbf{S}^j = \mathbf{S}^{j-1} \cup \{s_j\}$ and thus $\mathbf{S}^{j-1} \subset \mathbf{S}^j$. Furthermore, as hardness and representativeness scores are irrelevant to the selection process (*i.e.*, $h_{s_{j+1}}^j = h_{s_{j+1}}^{j+1}$, $r_{s_{j+1}}^j = r_{s_{j+1}}^{j+1}$), we have

$$v_{s_{j+1}}^j = h_{s_{j+1}}^j + r_{s_{j+1}}^j + d_{s_{j+1}}^j \geq h_{s_{j+1}}^{j+1} + r_{s_{j+1}}^{j+1} + d_{s_{j+1}}^{j+1} = v_{s_{j+1}}^{j+1}. \tag{9}$$

Finally, by combining Eq. (7) and Eq. (9), we arrive at

$$v_{s_j}^j \geq v_{s_{j+1}}^j \geq v_{s_{j+1}}^{j+1}, \tag{10}$$

which completes the proof of Theorem 1. $\qquad\square$

Upon selecting the most valuable samples, we inquire about their cluster affiliations relative to the nearest cluster centers. For each selected sample, we provide $T = 5$ nearest cluster center candidates [2], and then request the user to determine which candidate shares the same semantics with the anchor as illustrated in Fig. 2. Notably, such an inquiry strategy is more user-friendly than directly asking about the pair-wise correlation between two samples, by aiding users in grasping cluster semantics and partitioning criteria. User feedback could be either positive (selecting a candidate) or negative (rejecting all candidates) to the given sample, which serves the subsequent model finetuning strategy introduced in the next section.

## 3.2 Model Optimization

Based on the user feedback, we present a positive loss $\mathcal{L}_{pos}$, a negative loss $\mathcal{L}_{neg}$, and a regularization loss $\mathcal{L}_{reg}$ to finetune the clustering model:

$$\mathcal{L} = \mathcal{L}_{pos} + \mathcal{L}_{neg} + \mathcal{L}_{reg}. \tag{11}$$

The three loss terms are designed to utilize positive feedback, to use negative feedback, and to prevent overfitting the queried samples, respectively, with details provided below.

**Positive Loss.** Positive user feedback refers to identifying the cluster centroid sharing the same semantics with the inquiry sample. To exploit this feedback, we draw the sample and the cluster centroid closer by the following positive loss:

$$\mathcal{L}_{pos} = -\frac{1}{M_{pos}} \sum_{i=1}^{M_{pos}} \sum_{j=1}^{C} y_{ij} \log p_{ij}, \tag{12}$$

where $M_{pos}$ denotes the count of positive feedback, $C$ is the number of clusters, $p_{ij}$ refers to the probability of sample $i$ belonging to cluster $j$, and $y_{ij} \in \{0, 1\}$ is an indicator that equals one iff the $j$-th cluster is selected by user.

**Negative Loss.** Negative user feedback indicates that no candidates match the semantics of the inquiry sample. To leverage the feedback, we enforce the sample apart from all candidate clusters using the following negative loss:

$$\mathcal{L}_{neg} = -\frac{1}{M_{neg}} \sum_{i=1}^{M_{neg}} \sum_{j=1}^{C} \tilde{y}_{ij} \log(1 - p_{ij}), \tag{13}$$

where $M_{neg}$ is the count of negative feedback, and $\tilde{y}_{ij}$ is an indicator that equals one if the $j$-th cluster is the randomly chosen candidate, and zero otherwise.

**Regularization Loss.** To reduce the interaction cost, only a small amount of samples are selected for user interaction. However, exclusively finetuning the model with the above two losses risks overfitting to the inquiry samples, potentially compromising previously correct cluster predictions. To tackle this problem, we propose preserving the overall cluster boundary by retaining confident predictions. Formally, the regularization loss is defined as follows:

$$\mathcal{L}_{reg} = -\frac{1}{N} \sum_{i=1}^{N} \mathbb{1}[p_{i\hat{j}} > \tau] \log p_{i\hat{j}}, \ \ \hat{j} = \arg\max p_i \tag{14}$$

where $N$ is the count of all samples, $\tau = 0.99$ is the confidence threshold, $\mathbb{1}[\text{cond}] \in \{0, 1\}$ is an indicator that equals one iff the condition $\text{cond}$ holds.

The above three losses are applied to optimize the pre-trained clustering model for performance improvement. After finetuning, we could directly obtain the improved cluster assignments from the model's cluster head[3].

# 4 Experiments

In this section, we first apply the proposed IDC to two state-of-the-art deep clustering methods, and evaluate the performance on five widely used image clustering benchmarks. Then we conduct ablation studies and parameter analyses to validate the robustness and effectiveness of IDC.

## 4.1 Datasets and Evaluation Metrics

We evaluate IDC on five widely used image clustering datasets, including CIFAR-10 [17], CIFAR-20 [17], STL-10 [8], ImageNet-10 [5] and ImageNet-Dogs [5], as detailed in Table 1.

Three widely used clustering metrics are adopted for performance evaluation, including Normalized Mutual Information (NMI), Accuracy (ACC), and Adjusted Rand Index (ARI). Higher scores signify superior clustering results.

Table 1: A summary of the used datasets.

| Dataset | Split | Samples | Classes |
|---|---|---|---|
| CIFAR-10 | Train+Test | 60000 | 10 |
| CIFAR-20 | Train+Test | 60000 | 20 |
| STL-10 | Train+Test | 13000 | 10 |
| ImageNet-10 | Train | 13000 | 10 |
| ImageNet-Dogs | Train | 19500 | 15 |

## 4.2 Implementation Details

Without loss of generality, we apply the proposed IDC on two recent methods TCL [21] and ProPos [14], on behalf of deep clustering models with and without a cluster head, respectively. Notably, for clustering models without a cluster head like ProPos, we append a randomly initialized two-layer fully connected network as an alternative. In the model optimization stage, we finetune the pre-trained clustering model for $100$ epochs. For ProPos, we warm up the cluster head with the positive and negative loss in Eq. 12 and 13 in the first $50$ epochs, since the prediction confidences are unreliable initially. To balance the effect of user feedback and model regularization, we use two independent data loaders for the inquiry and confident samples, with batch sizes of $100$ and $500$, respectively. All images are augmented consistently with the pre-trained clustering model for finetuning, while the original images are used for value evaluation and pseudo-labeling. All experiments are conducted on a single Nvidia RTX 3090 GPU on the Ubuntu 20.04 platform with CUDA 12.0.

## 4.3 Comparisons with State of the Arts

We first compare the proposed IDC with $14$ recent deep clustering methods, including CC [18], SCAN [38], NMM [9], MiCE [37], BYOL [10], GCC [46], SPICE [29], IDFD [36], TCC [34], DivClust [26], SeCu [32], CoNR [45], TCL [21], and ProPos [14]. In addition, we include two representative semi-supervised classification and clustering baselines FixMatch [35] and Cop-Kmeans [40] for benchmarking. For FixMatch, we use the ResNet-34 [12] as the backbone and annotate the inquiry images with positive user feedback for fair comparisons. For Cop-Kmeans, we use the TCL image features as the input and transform the user feedback as must- and cannot-link constraints.

As shown in Table 2, IDC gains consistent performance improvement, especially on more challenging datasets. Specifically, IDC boosts the clustering accuracy of TCL/ProPos by $16.3\%/19.2\%$ and $14.4\%/9.2\%$ on CIFAR-20 and ImageNet-Dogs, respectively. Besides, the results show that solely correcting the cluster assignments of 500 samples brings marginal performance improvement, since they are only a small portion of the data. Notably, IDC also outperforms the semi-supervised baseline FixMatch. Such a result could be attributed to its customized valuable sample selection strategy. Namely, the selected inquiry samples are catered to the pre-trained clustering model, which may not suit the general semi-supervised classification. Moreover, the superior performance of IDC compared with Cop-Kmeans demonstrates its stronger ability to utilize user feedback for model optimization.

Table 2: Clustering performance comparison with the state-of-the-art methods on five benchmarks. The performance of $IDC_{ProPos}$ is unavailable as the code of ProPos on STL-10 has not been released. To make a clear comparison, we add a baseline by manually correcting the cluster assignments of 500 query samples, denoted by TCL[†] and ProPos[†].

| Method | CIFAR-10 | | | CIFAR-20 | | | STL-10 | | | ImageNet-10 | | | ImageNet-Dogs | | |
|---|---|---|---|---|---|---|---|---|---|---|---|---|---|---|---|
| | NMI | ACC | ARI | NMI | ACC | ARI | NMI | ACC | ARI | NMI | ACC | ARI | NMI | ACC | ARI |
| CC [18] | 70.5 | 79.0 | 63.7 | 43.1 | 42.9 | 26.6 | 76.4 | 85.0 | 72.6 | 85.9 | 89.3 | 82.2 | 44.5 | 42.9 | 27.4 |
| SCAN [38] | 79.7 | 88.3 | 77.2 | 48.6 | 50.7 | 33.3 | 69.8 | 80.9 | 64.6 | - | - | - | - | - | - |
| NMM [9] | 74.8 | 84.3 | 70.9 | 48.4 | 47.7 | 31.6 | 69.4 | 80.8 | 65.0 | - | - | - | - | - | - |
| MiCE [37] | 73.7 | 83.5 | 69.8 | 43.6 | 44.0 | 28.0 | 63.5 | 75.2 | 57.5 | - | - | - | 42.3 | 43.9 | 28.6 |
| BYOL [10] | 81.7 | 89.4 | 79.0 | 55.9 | 56.9 | 39.3 | 71.3 | 82.5 | 65.7 | 86.6 | 93.9 | 87.2 | 63.5 | 69.4 | 54.8 |
| GCC [46] | 76.4 | 85.6 | 72.8 | 47.2 | 47.2 | 30.5 | 68.4 | 78.8 | 63.1 | 84.2 | 90.1 | 82.2 | 49.0 | 52.6 | 36.2 |
| SPICE [29] | 73.4 | 83.8 | 70.5 | 44.8 | 46.8 | 29.4 | 81.7 | 90.8 | 81.2 | 82.8 | 92.1 | 83.6 | 57.2 | 64.6 | 47.9 |
| IDFD [36] | 71.1 | 81.5 | 66.3 | 42.6 | 42.5 | 26.4 | 64.3 | 75.6 | 57.5 | 89.8 | 95.4 | 90.1 | 54.6 | 59.1 | 41.3 |
| TCC [34] | 79.0 | 90.6 | 73.3 | 47.9 | 49.1 | 31.2 | 73.2 | 81.4 | 68.9 | 84.8 | 89.7 | 82.5 | 55.4 | 59.5 | 41.7 |
| DivClust [26] | 72.4 | 81.9 | 68.1 | 44.0 | 43.7 | 28.3 | - | - | - | 89.1 | 93.6 | 87.8 | 51.6 | 52.9 | 37.6 |
| SeCu [32] | 86.1 | 93.0 | 85.7 | 55.1 | 55.2 | 39.7 | 73.3 | 83.6 | 69.3 | - | - | - | - | - | - |
| CoNR [45] | 87.1 | 93.3 | 86.5 | 60.3 | 59.0 | 44.8 | 84.6 | 92.2 | 83.8 | 89.8 | 95.8 | 90.9 | 74.2 | 80.2 | 67.6 |
| FixMatch [35] | 86.8 | 92.8 | 85.4 | 57.2 | 67.2 | 47.3 | 61.7 | 68.6 | 49.2 | 84.2 | 92.5 | 84.4 | 50.0 | 57.9 | 33.7 |
| Cop-Kmeans [40] | 82.3 | 89.0 | 78.6 | 52.2 | 52.4 | 34.7 | 78.1 | 85.4 | 73.1 | 85.5 | 88.6 | 81.0 | 61.5 | 63.5 | 49.7 |
| TCL [21] | 81.9 | 88.7 | 78.0 | 52.9 | 53.1 | 35.7 | 79.9 | 86.8 | 75.7 | 87.5 | 89.5 | 83.7 | 62.3 | 64.4 | 51.6 |
| TCL[†] | 82.2 | 88.9 | 78.4 | 53.2 | 53.5 | 36.1 | 82.0 | 88.6 | 78.5 | 88.6 | 90.4 | 85.0 | 62.8 | 65.6 | 52.3 |
| **$IDC_{TCL}$(Ours)** | 84.4 | 92.7 | 84.8 | 58.1 | 69.4 | 48.7 | **85.3** | **92.7** | **84.6** | 93.2 | 97.2 | 93.9 | 69.1 | 78.8 | 63.6 |
| ProPos [14] | 87.7 | 93.6 | 87.1 | 59.1 | 59.1 | 43.6 | 75.8 | 86.7 | 73.7 | 88.9 | 95.2 | 89.6 | 73.0 | 76.9 | 66.9 |
| ProPos[†] | 87.9 | 93.7 | 87.3 | 59.3 | 59.4 | 43.8 | - | - | - | 89.6 | 95.5 | 90.3 | 73.8 | 77.8 | 67.8 |
| **$IDC_{ProPos}$(Ours)** | **90.5** | **95.7** | **90.9** | **69.2** | **78.3** | **61.4** | - | - | - | **93.2** | **97.3** | **94.1** | **77.6** | **86.1** | **74.8** |

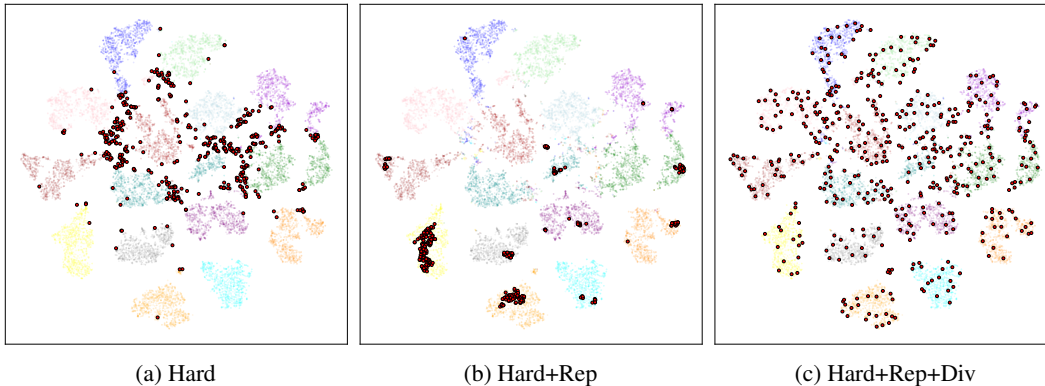

(a) Hard        (b) Hard+Rep        (c) Hard+Rep+Div

Figure 3: T-SNE visualizations of samples selected by different strategies among all data points, on the ImageNet-Dogs dataset where selected samples are highlighted by red dots.

## 4.4 Ablation Study and Parameter Analysis

To prove the robustness and effectiveness of IDC, we conduct ablation studies and parameter analyses on the TCL-based model. Specifically, for the user interaction stage, we study the effectiveness of the valuable sample selection strategy, as well as the impact of the number of selected samples $M$ and candidate cluster centers $T$. For the model optimization stage, we investigate the effectiveness of the three loss terms.

**Effectiveness of the valuable sample selection strategy.** As detailed in Section 3.1, starting with the clustering hardness, we additionally consider representativeness and diversity to quantify the value of each sample. Here, to provide an intuitive understanding of the three criteria, we visualize the selected samples among all data points in Fig. 3. As can be seen, solely considering hardness would select most boundary samples, which are not representative enough and thus sub-optimal for reducing

Table 3: Performance with different sample selection strategies on CIFAR-20 and ImageNet-Dogs.

| Selection Strategy | CIFAR-20 | | | ImageNet-Dogs | | |
|---|---|---|---|---|---|---|
| | NMI | ACC | ARI | NMI | ACC | ARI |
| None (Pre-trained Model) | 52.2 | 52.6 | 34.9 | 61.8 | 64.1 | 50.9 |
| Hard | 51.3 | 57.7 | 37.0 | 68.2 | 75.9 | 60.2 |
| Hard+Rep | 35.8 | 36.3 | 12.6 | 59.7 | 65.3 | 49.4 |
| Hard+Rep+Div | **58.1** | **69.4** | **48.7** | **69.1** | **78.8** | **63.6** |

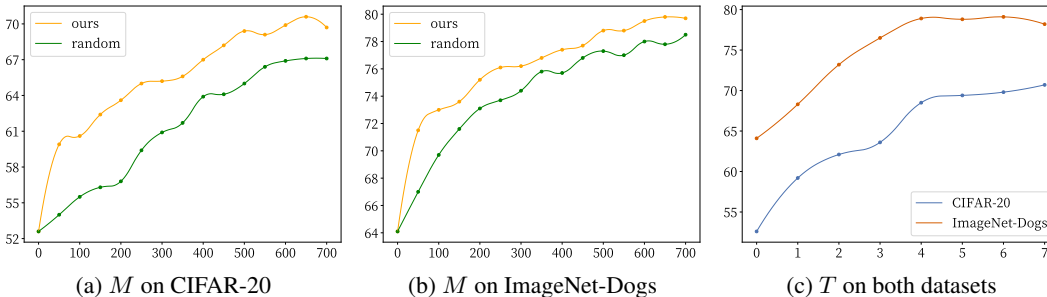

(a) $M$ on CIFAR-20   (b) $M$ on ImageNet-Dogs   (c) $T$ on both datasets

Figure 4: Influence of different numbers of selected samples $M$ and candidates $T$. (a)–(b) Clustering accuracy with different $M$ on CIFAR-20 and ImageNet-Dogs respectively, compared with the random selection baseline. (c) Clustering accuracy with different $T$ on both datasets.

the interaction cost. When additionally considering the representativeness, however, the selected samples would collapse into dense subsets, leading to a significant performance drop as shown in Table 3. Finally, further integrating the diversity results in samples simultaneously comprising the three expected characteristics, which gives the best clustering performance.

**Impact of the number of selected samples and candidates.** For interaction cost reduction, we select $M = 500$ most valuable samples for the user inquiry, and $T = 5$ nearest cluster centers as the candidates. Here, we investigate how different numbers of $M$ and $T$ influence the final clustering performance. As depicted in Fig. 4 (a) and (b), the performance of IDC improves as the number of selected samples increases. Notably, the improvement grows rapidly at the start but gradually levels off as more samples are selected. Moreover, valuable samples selected by IDC consistently outperform the random selection baseline. These results not only demonstrate the effectiveness of our valuable sample selection strategy, but also verify the monotonously decreased sample value as proved in Theorem 1. For the number of candidates, Fig. 4 (c) shows that comparing the inquiry sample with the five nearest centers strikes the best balance between performance and interaction cost.

Table 4: Performance with different combinations of loss terms on CIFAR-20 and ImageNet-Dogs.

| $\mathcal{L}_{pos}$ | $\mathcal{L}_{neg}$ | $\mathcal{L}_{reg}$ | CIFAR-20 | | | ImageNet-Dogs | | |
|---|---|---|---|---|---|---|---|---|
| | | | NMI | ACC | ARI | NMI | ACC | ARI |
| ✓ | | | 56.8 | 65.8 | 46.2 | 68.3 | 76.9 | 62.5 |
| | ✓ | | 7.0 | 9.6 | 2.4 | 26.8 | 24.1 | 3.1 |
| | | ✓ | 50.9 | 52.9 | 35.2 | 61.7 | 64.9 | 51.8 |
| ✓ | ✓ | | 55.0 | 66.2 | 44.1 | 67.9 | 77.5 | 61.6 |
| ✓ | | ✓ | **58.7** | 67.6 | 47.8 | 68.8 | 77.5 | 62.7 |
| | ✓ | ✓ | 36.7 | 39.2 | 17.3 | 53.1 | 59.7 | 34.8 |
| ✓ | ✓ | ✓ | 58.1 | **69.4** | **48.7** | **69.1** | **78.8** | **63.6** |

**Effectiveness of the loss terms.** To prove the effectiveness of the positive loss $\mathcal{L}_{pos}$ in Eq. (12), the negative loss $\mathcal{L}_{neg}$ in Eq. (13), and the regularization loss in Eq. (14), we evaluate different

combination of the three losses and the results are shown in Fig. 4. On the one hand, no single loss is adequate to yield promising clustering results. In particular, solely leveraging the negative loss would deny all the Top-5 predictions and thus severely damage the decision boundary of the pre-trained clustering model, leading to the model collapse. On the other hand, each loss is indispensable during the model optimization. Notably, the positive loss brings the most substantial performance improvement, as it offers the most direct clustering guidance to the model.

## 5 Conclusion

Instead of mining semantics from internal data, we propose an interactive deep clustering method IDC, which incorporates user interaction to address the hard sample problem. By mathematically measuring the sample value defined on hardness, representativeness, and diversity, IDC selects the highest-value samples and inquiries about their cluster affiliations through a user-friendly interaction interface. By fine-tuning the pre-trained clustering model leveraging user feedback, IDC remarkably improves the performance of various state-of-the-art deep clustering methods. For future studies, one potential direction is to consider the mistakes in user feedback, and correspondingly improve the robustness of IDC. Another possible direction is to design a more advanced interaction pipeline, for aligning clustering results with the user's personalized partition criterion. In general, we hope this work could provide novel insight to the community, attracting more attention to the interactive clustering paradigm which is a promising and less explored area.

## Acknowledgements

This work was supported in part by NSFC under Grant 62176171, U21B2040, 623B2075, 62472295; in part by the Fundamental Research Funds for the Central Universities under Grant CJ202303; and in part by Sichuan Science and Technology Planning Project under Grant 24NSFTD0130.

## Footnotes

[2]In practice, each cluster center is represented by its nearest sample.

[3]We append a cluster head for pre-trained clustering models that do not have one. More details are provided in Section 4.2.

## References

[1] Henrique Aguiar, Mauro Santos, Peter Watkinson, and Tingting Zhu. Learning of cluster-based feature importance for electronic health record time-series. In *International conference on machine learning*, pages 161–179. PMLR, 2022.

[2] Edward Ayers, Jonathan Sadeghi, John Redford, Romain Mueller, and Puneet K Dokania. Query-based hard-image retrieval for object detection at test time. In *Proceedings of the AAAI Conference on Artificial Intelligence*, volume 37, pages 14692–14700, 2023.

[3] Sugato Basu, Arindam Banerjee, and Raymond J. Mooney. Semi-supervised clustering by seeding. In *Proceedings of the Nineteenth International Conference on Machine Learning*, ICML '02, page 27–34, San Francisco, CA, USA, 2002. Morgan Kaufmann Publishers Inc.

[4] Shaotian Cai, Liping Qiu, Xiaojun Chen, Qin Zhang, and Longteng Chen. Semantic-enhanced image clustering. In *Proceedings of the AAAI conference on artificial intelligence*, volume 37, pages 6869–6878, 2023.

[5] Jianlong Chang, Lingfeng Wang, Gaofeng Meng, Shiming Xiang, and Chunhong Pan. Deep adaptive image clustering. In *Proceedings of the IEEE international conference on computer vision*, pages 5879–5887, 2017.

[6] Ting Chen, Simon Kornblith, Mohammad Norouzi, and Geoffrey Hinton. A simple framework for contrastive learning of visual representations. In *International conference on machine learning*, pages 1597–1607. PMLR, 2020.

[7] Jang Hyun Cho, Utkarsh Mall, Kavita Bala, and Bharath Hariharan. Picie: Unsupervised semantic segmentation using invariance and equivariance in clustering. In *Proceedings of the IEEE/CVF Conference on Computer Vision and Pattern Recognition*, pages 16794–16804, 2021.

[8] Adam Coates, Andrew Ng, and Honglak Lee. An analysis of single-layer networks in unsupervised feature learning. In *Proceedings of the fourteenth international conference on artificial intelligence and statistics*, pages 215–223. JMLR Workshop and Conference Proceedings, 2011.

[9] Zhiyuan Dang, Cheng Deng, Xu Yang, Kun Wei, and Heng Huang. Nearest neighbor matching for deep clustering. In *Proceedings of the IEEE/CVF conference on computer vision and pattern recognition*, pages 13693–13702, 2021.

[10] Jean-Bastien Grill, Florian Strub, Florent Altché, Corentin Tallec, Pierre Richemond, Elena Buchatskaya, Carl Doersch, Bernardo Avila Pires, Zhaohan Guo, Mohammad Gheshlaghi Azar, et al. Bootstrap your own latent-a new approach to self-supervised learning. *Advances in neural information processing systems*, 33:21271–21284, 2020.

[11] Kaiming He, Haoqi Fan, Yuxin Wu, Saining Xie, and Ross Girshick. Momentum contrast for unsupervised visual representation learning. In *Proceedings of the IEEE/CVF conference on computer vision and pattern recognition*, pages 9729–9738, 2020.

[12] Kaiming He, Xiangyu Zhang, Shaoqing Ren, and Jian Sun. Deep residual learning for image recognition. In *Proceedings of the IEEE conference on computer vision and pattern recognition*, pages 770–778, 2016.

[13] Jiabo Huang, Shaogang Gong, and Xiatian Zhu. Deep semantic clustering by partition confidence maximisation. In *Proceedings of the IEEE/CVF conference on computer vision and pattern recognition*, pages 8849–8858, 2020.

[14] Zhizhong Huang, Jie Chen, Junping Zhang, and Hongming Shan. Learning representation for clustering via prototype scattering and positive sampling. *IEEE Transactions on Pattern Analysis and Machine Intelligence*, 2022.

[15] Xu Ji, Joao F Henriques, and Andrea Vedaldi. Invariant information clustering for unsupervised image classification and segmentation. In *Proceedings of the IEEE/CVF international conference on computer vision*, pages 9865–9874, 2019.

[16] Zhen Jiang, Yongzhao Zhan, Qirong Mao, and Yang Du. Semi-supervised clustering under a "compact-cluster" assumption. *IEEE Transactions on Knowledge and Data Engineering*, 35(5):5244–5256, 2022.

[17] Alex Krizhevsky, Geoffrey Hinton, et al. Learning multiple layers of features from tiny images. 2009.

[18] Yunfan Li, Peng Hu, Zitao Liu, Dezhong Peng, Joey Tianyi Zhou, and Xi Peng. Contrastive clustering. In *Proceedings of the AAAI conference on artificial intelligence*, volume 35, pages 8547–8555, 2021.

[19] Yunfan Li, Peng Hu, Dezhong Peng, Jiancheng Lv, Jianping Fan, and Xi Peng. Image clustering with external guidance. *arXiv preprint arXiv:2310.11989*, 2023.

[20] Yunfan Li, Yijie Lin, Peng Hu, Dezhong Peng, Han Luo, and Xi Peng. Single-cell rna-seq debiased clustering via batch effect disentanglement. *IEEE Transactions on Neural Networks and Learning Systems*, pages 1–11, 2023.

[21] Yunfan Li, Mouxing Yang, Dezhong Peng, Taihao Li, Jiantao Huang, and Xi Peng. Twin contrastive learning for online clustering. *International Journal of Computer Vision*, 130(9):2205–2221, 2022.

[22] Sihang Liu, Wenming Cao, Ruigang Fu, Kaixiang Yang, and Zhiwen Yu. Rpsc: Robust pseudo-labeling for semantic clustering. In *Proceedings of the AAAI Conference on Artificial Intelligence*, volume 38, pages 14008–14016, 2024.

[23] Xinran Ma, Mouxing Yang, Yunfan Li, Peng Hu, Jiancheng Lv, and Xi Peng. Cross-modal retrieval with noisy correspondence via consistency refining and mining. *IEEE transactions on image processing*, 2024.

[24] Laura Manduchi, Kieran Chin-Cheong, Holger Michel, Sven Wellmann, and Julia Vogt. Deep conditional gaussian mixture model for constrained clustering. *Advances in Neural Information Processing Systems*, 34:11303–11314, 2021.

[25] Amir Markovitz, Gilad Sharir, Itamar Friedman, Lihi Zelnik-Manor, and Shai Avidan. Graph embedded pose clustering for anomaly detection. In *Proceedings of the IEEE/CVF Conference on Computer Vision and Pattern Recognition*, pages 10539–10547, 2020.

[26] Ioannis Maniadis Metaxas, Georgios Tzimiropoulos, and Ioannis Patras. Divclust: Controlling diversity in deep clustering. In *Proceedings of the IEEE/CVF Conference on Computer Vision and Pattern Recognition*, pages 3418–3428, 2023.

[27] Tri Nguyen, Shahana Ibrahim, and Xiao Fu. Deep clustering with incomplete noisy pairwise annotations: A geometric regularization approach. In *International Conference on Machine Learning*, pages 25980–26007. PMLR, 2023.

[28] Dong Nie, Li Wang, Lei Xiang, Sihang Zhou, Ehsan Adeli, and Dinggang Shen. Difficulty-aware attention network with confidence learning for medical image segmentation. In *Proceedings of the AAAI Conference on Artificial Intelligence*, volume 33, pages 1085–1092, 2019.

[29] Chuang Niu, Hongming Shan, and Ge Wang. Spice: Semantic pseudo-labeling for image clustering. *IEEE Transactions on Image Processing*, 31:7264–7278, 2022.

[30] Xi Peng, Jiashi Feng, Shijie Xiao, Wei-Yun Yau, Joey Tianyi Zhou, and Songfan Yang. Structured autoencoders for subspace clustering. *IEEE Transactions on Image Processing*, 27(10):5076–5086, 2018.

[31] Xi Peng, Shijie Xiao, Jiashi Feng, Wei-Yun Yau, and Zhang Yi. Deep subspace clustering with sparsity prior. In *IJCAI*, pages 1925–1931, 2016.

[32] Qi Qian. Stable cluster discrimination for deep clustering. In *Proceedings of the IEEE/CVF International Conference on Computer Vision*, pages 16645–16654, 2023.

[33] Florian Schroff, Dmitry Kalenichenko, and James Philbin. Facenet: A unified embedding for face recognition and clustering. In *Proceedings of the IEEE conference on computer vision and pattern recognition*, pages 815–823, 2015.

[34] Yuming Shen, Ziyi Shen, Menghan Wang, Jie Qin, Philip Torr, and Ling Shao. You never cluster alone. *Advances in Neural Information Processing Systems*, 34:27734–27746, 2021.

[35] Kihyuk Sohn, David Berthelot, Nicholas Carlini, Zizhao Zhang, Han Zhang, Colin A Raffel, Ekin Dogus Cubuk, Alexey Kurakin, and Chun-Liang Li. Fixmatch: Simplifying semi-supervised learning with consistency and confidence. *Advances in neural information processing systems*, 33:596–608, 2020.

[36] Yaling Tao, Kentaro Takagi, and Kouta Nakata. Clustering-friendly representation learning via instance discrimination and feature decorrelation. *arXiv preprint arXiv:2106.00131*, 2021.

[37] Tsung Wei Tsai, Chongxuan Li, and Jun Zhu. Mice: Mixture of contrastive experts for unsupervised image clustering. In *International conference on learning representations*, 2020.

[38] Wouter Van Gansbeke, Simon Vandenhende, Stamatios Georgoulis, Marc Proesmans, and Luc Van Gool. Scan: Learning to classify images without labels. In *European conference on computer vision*, pages 268–285. Springer, 2020.

[39] Kiri Wagstaff and Claire Cardie. Clustering with instance-level constraints. *AAAI/IAAI*, 1097:577–584, 2000.

[40] Kiri Wagstaff, Claire Cardie, Seth Rogers, Stefan Schrödl, et al. Constrained k-means clustering with background knowledge. In *Icml*, volume 1, pages 577–584, 2001.

[41] Jianlong Wu, Keyu Long, Fei Wang, Chen Qian, Cheng Li, Zhouchen Lin, and Hongbin Zha. Deep comprehensive correlation mining for image clustering. In *Proceedings of the IEEE/CVF international conference on computer vision*, pages 8150–8159, 2019.

[42] Junyuan Xie, Ross Girshick, and Ali Farhadi. Unsupervised deep embedding for clustering analysis. In *International conference on machine learning*, pages 478–487. PMLR, 2016.

[43] Jianwei Yang, Devi Parikh, and Dhruv Batra. Joint unsupervised learning of deep representations and image clusters. In *Proceedings of the IEEE conference on computer vision and pattern recognition*, pages 5147–5156, 2016.

[44] Mouxing Yang, Zhenyu Huang, Peng Hu, Taihao Li, Jiancheng Lv, and Xi Peng. Learning with twin noisy labels for visible-infrared person re-identification. In *Proceedings of the IEEE/CVF conference on computer vision and pattern recognition*, pages 14308–14317, 2022.

[45] Chunlin Yu, Ye Shi, and Jingya Wang. Contextually affinitive neighborhood refinery for deep clustering. *Advances in Neural Information Processing Systems*, 36, 2024.

[46] Huasong Zhong, Jianlong Wu, Chong Chen, Jianqiang Huang, Minghua Deng, Liqiang Nie, Zhouchen Lin, and Xian-Sheng Hua. Graph contrastive clustering. In *Proceedings of the IEEE/CVF international conference on computer vision*, pages 9224–9233, 2021.

